# Semi-supervised Learning via Gaussian Processes

**Neil D. Lawrence**
Department of Computer Science
University of Sheffield
Sheffield, S1 4DP, U.K.
`neil@dcs.shef.ac.uk`

**Michael I. Jordan**
Computer Science and Statistics
University of California
Berkeley, CA 94720, U.S.A.
`jordan@cs.berkeley.edu`

## Abstract

We present a probabilistic approach to learning a Gaussian Process classifier in the presence of unlabeled data. Our approach involves a "null category noise model" (NCNM) inspired by ordered categorical noise models. The noise model reflects an assumption that the data density is lower between the class-conditional densities. We illustrate our approach on a toy problem and present comparative results for the semi-supervised classification of handwritten digits.

## 1 Introduction

The traditional machine learning classification problem involves a set of input vectors $\mathbf{X} = [\mathbf{x}_1 \ldots \mathbf{x}_N]^{\mathrm{T}}$ and associated labels $\mathbf{y} = [y_1 \ldots y_N]^{\mathrm{T}}$, $y_n \in \{-1, 1\}$. The goal is to find a mapping between the inputs and the labels that yields high predictive accuracy. It is natural to consider whether such predictive performance can be improved via "semi-supervised learning," in which a combination of labeled data and unlabeled data are available.

Probabilistic approaches to classification either estimate the class-conditional densities or attempt to model $p(y_n|\mathbf{x}_n)$ directly. In the latter case, if we fail to make any assumptions about the underlying distribution of input data, the unlabeled data will not affect our predictions. Thus, most attempts to make use of unlabeled data within a probabilistic framework focus on incorporating a model of $p(\mathbf{x}_n)$: for example, by treating it as a mixture, $\sum_{y_n} p(\mathbf{x}_n|y_n) p(y_n)$, and inferring $p(y_n|\mathbf{x}_n)$ (e.g., [5]), or by building kernels based on $p(\mathbf{x}_n)$ (e.g., [8]). These approaches can be unwieldy, however, in that the complexities of the input distribution are typically of little interest when performing classification, so that much of the effort spent modelling $p(\mathbf{x}_n)$ may be wasted.

An alternative is to make weaker assumptions regarding $p(\mathbf{x}_n)$ that are of particular relevance to classification. In particular, the *cluster assumption* asserts that the data density should be reduced in the vicinity of a decision boundary (e.g., [2]). Such a qualitative assumption is readily implemented within the context of non-probabilistic kernel-based classifiers. In the current paper we take up the challenge

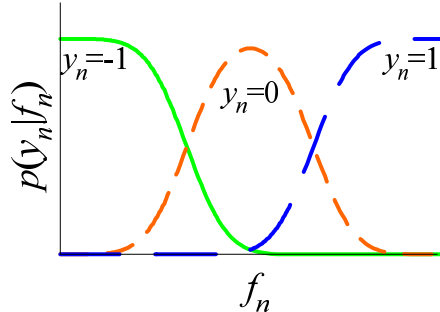

Figure 1: The ordered categorical noise model. The plot shows $p(y_n|f_n)$ for different values of $y_n$. Here we have assumed three categories.

of showing how it can be achieved within a (nonparametric) probabilistic framework.

Our approach involves a notion of a "null category region," a region which acts to exclude unlabeled data points. Such a region is analogous to the traditional notion of a "margin" and indeed our approach is similar in spirit to the transductive SVM [10], which seeks to maximize the margin by allocating labels to the unlabeled data. A major difference, however, is that our approach maintains and updates the process variance (not merely the process mean) and, as we will see, this variance turns out to interact in a significant way with the null category concept.

The structure of the paper is as follows. We introduce the basic probabilistic framework in Section 2 and discuss the effect of the null category in Section 3. Section 4 discusses posterior process updates and prediction. We present comparative experimental results in Section 5 and present our conclusions in Section 6.

## 2 Probabilistic Model

In addition to the input vector $\mathbf{x}_n$ and the label $y_n$, our model includes a latent process variable $f_n$, such that the probability of class membership decomposes as $p(y_n|\mathbf{x}_n) = \int p(y_n|f_n)\, p(f_n|\mathbf{x}_n)\, df_n$. We first focus on the *noise model*, $p(y_n|f_n)$, deferring the discussion of an appropriate *process model*, $p(f_n|\mathbf{x}_n)$, to later.

### 2.1 Ordered categorical models

We introduce a novel noise model which we have termed a *null category noise model*, as it derives from the general class of *ordered categorical models* [1]. In the specific context of binary classification, our focus in this paper, we consider an ordered categorical model containing three categories[1].

$$p(y_n|f_n) = \begin{cases} \phi\left(-\left(f_n + \frac{w}{2}\right)\right) & \text{for } y_n = -1 \\ \phi\left(f_n + \frac{w}{2}\right) - \phi\left(f_n - \frac{w}{2}\right) & \text{for } y_n = 0 \\ \phi\left(f_n - \frac{w}{2}\right) & \text{for } y_n = 1 \end{cases},$$

where $\phi(x) = \int_{-\infty}^{x} N(z|0,1)\, dz$ is the cumulative Gaussian distribution function and $w$ is a parameter giving the width of category $y_n = 0$ (see Figure 1). We can also express this model in an equivalent and simpler form by replacing the

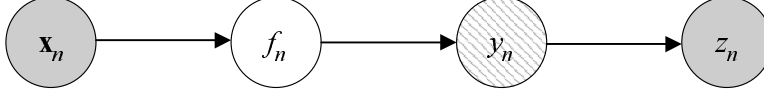

Figure 2: Graphical representation of the null category model. The fully-shaded nodes are always observed, whereas the lightly-shaded node is observed when $z_n = 0$.

cumulative Gaussian distribution by a Heaviside step function $H(\cdot)$ and adding independent Gaussian noise to the process model:

$$p\left(y_n|f_n\right) = \begin{cases} H\left(-\left(f_n + \frac{1}{2}\right)\right) & \text{for } y_n = -1 \\ H\left(f_n + \frac{1}{2}\right) - H\left(f_n - \frac{1}{2}\right) & \text{for } y_n = 0 \\ H\left(f_n - \frac{1}{2}\right) & \text{for } y_n = 1 \end{cases},$$

where we have standardized the width parameter to 1, by assuming that the overall scale is also handled by the process model.

To use this model in an unlabeled setting we introduce a further variable, $z_n$, which is one if a data point is unlabeled and zero otherwise. We first impose

$$p\left(z_n = 1|y_n = 0\right) = 0; \tag{1}$$

in other words, a data point can not be from the category $y_n = 0$ and be unlabeled. We assign probabilities of missing labels to the other classes $p\left(z_n = 1|y_n = 1\right) = \gamma_+$ and $p\left(z_n = 1|y_n = -1\right) = \gamma_-$. We see from the graphical representation in Figure 2 that $z_n$ is $d$-separated from $\mathbf{x}_n$. Thus when $y_n$ is observed, the posterior process is updated by using $p\left(y_n|f_n\right)$. On the other hand, when the data point is unlabeled the posterior process must be updated by $p\left(z_n|f_n\right)$ which is easily computed as:

$$p\left(z_n = 1|f_n\right) = \sum_{y_n} p\left(y_n|f_n\right) p\left(z_n = 1|y_n\right).$$

The "effective likelihood function" for a single data point, $L\left(f_n\right)$, therefore takes one of three forms:

$$L\left(f_n\right) = \begin{cases} H\left(-\left(f_n + \frac{1}{2}\right)\right) & \text{for} & y_n = -1, z_n = 0 \\ \gamma_- H\left(-\left(f_n + \frac{1}{2}\right)\right) + \gamma_+ H\left(f_n - \frac{1}{2}\right) & \text{for} & z_n = 1 \\ H\left(f_n - \frac{1}{2}\right) & \text{for} & y_n = 1\, z_n = 0 \end{cases}.$$

The constraint imposed by (1) implies that an unlabeled data point never comes from the class $y_n = 0$. Since $y_n = 0$ lies between the labeled classes this is equivalent to a hard assumption that no data comes from the region around the decision boundary. We can also soften this hard assumption if so desired by injection of noise into the process model. If we also assume that our labeled data only comes from the classes $y_n = 1$ and $y_n = -1$ we will never obtain any evidence for data with $y_n = 0$; for this reason we refer to this category as the *null category* and the overall model as a *null category noise model* (NCNM).

## 3  Process Model and Effect of the Null Category

We work within the Gaussian process framework and assume

$$p\left(f_n|\mathbf{x}_n\right) = N\left(f_n|\mu\left(\mathbf{x}_n\right), \varsigma\left(\mathbf{x}_n\right)\right),$$

where the mean $\mu\left(\mathbf{x}_n\right)$ and the variance $\varsigma\left(\mathbf{x}_n\right)$ are functions of the input space. A natural consideration in this setting is the effect of our likelihood function on the

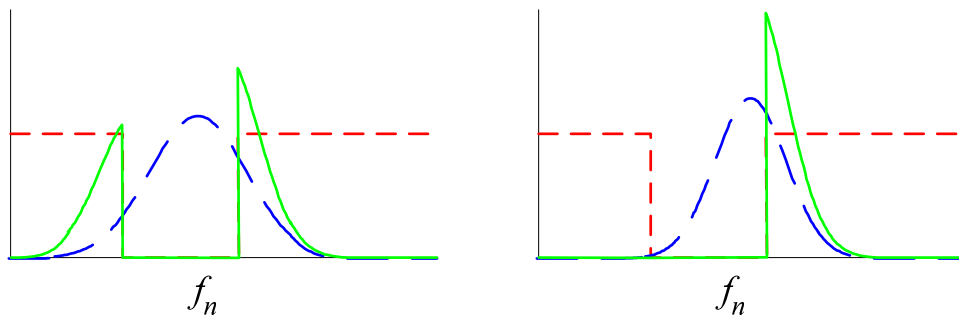

Figure 3: Two situations of interest. Diagrams show the prior distribution over $f_n$ (long dashes) the effective likelihood function from the noise model when $z_n = 1$ (short dashes) and a schematic of the resulting posterior over $f_n$ (solid line). *Left*: The posterior is bimodal and has a larger variance than the prior. *Right*: The posterior has one dominant mode and a lower variance than the prior. In both cases the process is pushed away from the null category.

distribution over $f_n$ from incorporating a new data point. First we note that if $y_n \in \{-1, 1\}$ the effect of the likelihood will be similar to that incurred in binary classification, in that the posterior will be a convolution of the step function and a Gaussian distribution. This is comforting as when a data point is labeled the model will act in a similar manner to a standard binary classification model. Consider now the case when the data point is unlabeled. The effect will depend on the mean and variance of $p(f_n|\mathbf{x}_n)$. If this Gaussian has little mass in the null category region, the posterior will be similar to the prior. However, if the Gaussian has significant mass in the null category region, the outcome may be loosely described in two ways:

1. If $p(f_n|\mathbf{x}_n)$ "spans the likelihood," Figure 3 (Left), then the mass of the posterior can be apportioned to either side of the null category region, leading to a bimodal posterior. The variance of the posterior will be greater than the variance of the prior, a consequence of the fact that the effective likelihood function is not log-concave (as can be easily verified).

2. If $p(f_n|\mathbf{x}_n)$ is "rectified by the likelihood," Figure 3 (Right), then the mass of the posterior will be pushed in to one side of the null category and the variance of the posterior will be smaller than the variance of the prior.

Note that for all situations when a portion of the mass of the prior distribution falls within the null category region it is pushed out to one side or both sides. The intuition behind the two situations is that in case 1, it is not clear what label the data point has, however it is clear that it shouldn't be where it currently is (in the null category). The result is that the process variance increases. In case 2 the data point is being assigned a label and the decision boundary is pushed to one side of the point so that it is classified according to the assigned label.

## 4   Posterior Inference and Prediction

Broadly speaking the effects discussed above are independent of the process model: the effective likelihood will always force the latent function away from the null category. To implement our model, however, we must choose a process model and an inference method. The nature of the noise model means that it is unlikely that we will find a non-trivial process model for which inference (in terms of marginalizing

$f_n$) will be tractable. We therefore turn to approximations which are inspired by "assumed density filtering" (ADF) methods; see, e.g., [3]. The idea in ADF is to approximate the (generally non-Gaussian) posterior with a Gaussian by matching the moments between the approximation and the true posterior. ADF has also been extended to allow each approximation to be revisited and improved as the posterior distribution evolves [7].

Recall from Section 3 that the noise model is not log-concave. When the variance of the process increases the best Gaussian approximation to our noise model can have negative variance. This situation is discussed in [7], where various suggestions are given to cope with the issue. In our implementation we followed the simplest suggestion: we set a negative variance to zero.

One important advantage of the Gaussian process framework is that hyperparameters in the covariance function (i.e., the kernel function), can be optimized by type-II maximum likelihood. In practice, however, if the process variance is maximized in an unconstrained manner the effective width of the null category can be driven to zero, yielding a model that is equivalent to a standard binary classification noise model[2]. To prevent this from happening we regularize with an L1 penalty on the process variances (this is equivalent to placing an exponential prior on those parameters).

## 4.1  Prediction with the NCNM

Once the parameters of the process model have been learned, we wish to make predictions about a new test-point $\mathbf{x}_*$ via the marginal distribution $p(y_*|\mathbf{x}_*)$. For the NCNM an issue arises here: this distribution will have a non-zero probability of $y_* = 0$, a label that does not exist in either our labeled or unlabeled data. This is where the role of $z$ becomes essential. The new point also has $z_* = 1$ so in reality the probability that a data point is from the positive class is given by

$$p(y_*|\mathbf{x}_*, z_*) \propto p(z_*|y_*) \, p(y_*|\mathbf{x}_*). \qquad (2)$$

The constraint that $p(z_*|y_* = 0) = 0$ causes the predictions to be correctly normalized. So for the distribution to be correctly normalized for a test data point we must assume that we have observed $z_* = 1$.

An interesting consequence is that observing $\mathbf{x}_*$ will have an effect on the process model. This is contrary to the standard Gaussian process setup (see, e.g., [11]) in which the predictive distribution depends only on the labeled training data and the location of the test point $\mathbf{x}_*$. In the NCNM the entire process model $p(f_*|\mathbf{x}_*)$ should be updated after the observation of $\mathbf{x}_*$. This is not a particular disadvantage of our approach; rather, it is an inevitable consequence of any method that allows unlabeled data to affect the location of the decision boundary—a consequence that our framework makes explicit. In our experiments, however, we disregard such considerations and make (possibly suboptimal) predictions of the class labels according to (2).

## 5  Experiments

Sparse representations of the data set are essential for speeding up the process of learning. We made use of the informative vector machine[3] (IVM) approach [6] to

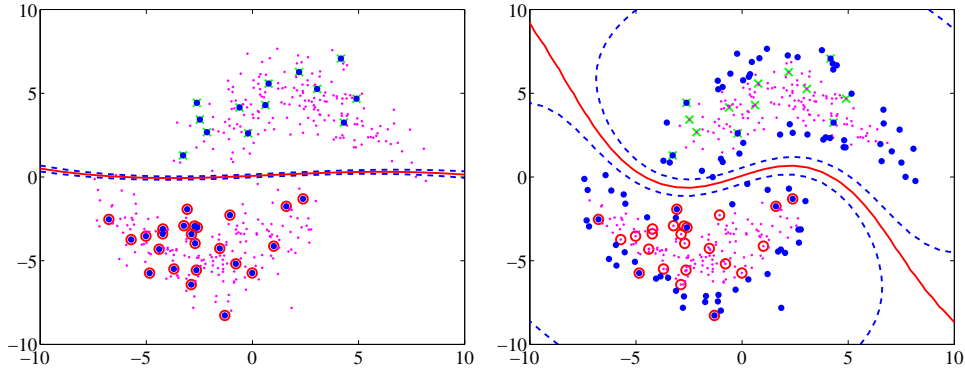

Figure 4: Results from the toy problem. There are 400 points, which are labeled with probability 0.1. Labelled data-points are shown as circles and crosses. Data-points in the active set are shown as large dots. All other data-points are shown as small dots. *Left*: Learning on the labeled data only with the IVM algorithm. All labeled points are used in the active set. *Right*: Learning on the labeled and unlabeled data with the NCNM. There are 100 points in the active set. In both plots decision boundaries are shown as a solid line; dotted lines represent contours within 0.5 of the decision boundary (for the NCNM this is the edge of the null category).

greedily select an active set according to information-theoretic criteria. The IVM also enables efficient learning of kernel hyperparameters, and we made use of this feature in all of our experiments. In all our experiments we used a kernel of the form

$$k_{nm} = \theta_2 \exp\left(-\theta_1 \left(\mathbf{x}_n - \mathbf{x}_m\right)^{\mathrm{T}} \left(\mathbf{x}_n - \mathbf{x}_m\right)\right) + \theta_3 \delta_{nm},$$

where $\delta_{nm}$ is the Kronecker delta function. The IVM algorithm selects an active set, and the parameters of the kernel were learned by performing type-II maximum likelihood over the active set. Since active set selection causes the marginalized likelihood to fluctuate it cannot be used to monitor convergence, we therefore simply iterated fifteen times between active set selection and kernel parameter optimisation. The parameters of the noise model, $\{\gamma_+, \gamma_-\}$ can also be optimized, but note that if we constrain $\gamma_+ = \gamma_- = \gamma$ then the likelihood is maximized by setting $\gamma$ to the proportion of the training set that is unlabeled.

We first considered an illustrative toy problem to demonstrate the capabilities of our model. We generated two-dimensional data in which two class-conditional densities interlock. There were 400 points in the original data set. Each point was labeled with probability 0.1, leading to 37 labeled points. First a standard IVM classifier was trained on the labeled data only (Figure 4, Left). We then used the null category approach to train a classifier that incorporates the unlabeled data. As shown in Figure 4 (Right), the resulting decision boundary finds a region of low data density and more accurately reflects the underlying data distribution.

## 5.1 High-dimensional example

To explore the capabilities of the model when the data set is of a much higher dimensionality we considered the USPS data set[4] of handwritten digits. The task chosen was to separate the digit 3 from 5. To investigate performance across a range of different operating conditions, we varied the proportion of unlabeled data between

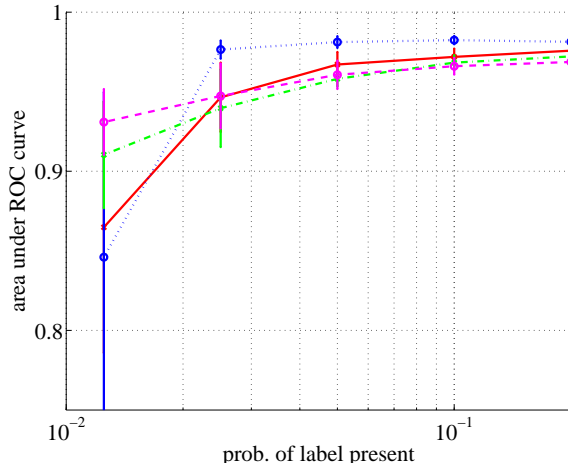

Figure 5: Area under the ROC curve plotted against probability of a point being labeled. Mean and standard errors are shown for the IVM (solid line), the NCNM (dotted line), the SVM (dash-dot line) and the transductive SVM (dashed line).

$0.2$ and $1.25 \times 10^{-2}$. We compared four classifiers: a standard IVM trained on the labeled data only, a support vector machine (SVM) trained on the labeled data only, the NCNM trained on the combined labeled-unlabeled data, and an implementation of the transductive SVM trained on the combined labeled-unlabeled data. The SVM and transductive SVM used the SVM$^{light}$ software [4]. For the SVM, the kernel inverse width hyperparameter $\theta_1$ was set to the value learned by the IVM. For the transductive SVM it was set to the higher of the two values learned by the IVM and the NCNM[5]. For the SVM-based models we set $\theta_2 = 1$ and $\theta_3 = 0$; the margin error cost, $C$, was left at the SVM$^{light}$ default setting.

The quality of the resulting classifiers was evaluated by computing the area under the ROC curve for a previously unseen test data set. Each run was completed ten times with different random seeds. The results are summarized in Figure 5.

The results show that below a label probability of $2.5 \times 10^{-2}$ both the SVM and transductive SVM outperform the NCNM. In this region the estimate $\theta_1$ provided by the NCNM was sometimes very low leading to occasional very poor results (note the large error bar). Above $2.5 \times 10^{-2}$ a clear improvement is obtained for the NCNM over the other models. It is of interest to contrast this result with an analogous experiment on discriminating twos vs. threes in [8], where $p(\mathbf{x}_n)$ was used to derive a kernel. No improvement was found in this case, which [8] attributed to the difficulties of modelling $p(\mathbf{x}_n)$ in high dimensions. These difficulties appear to be diminished for the NCNM, presumably because it never explicitly models $p(\mathbf{x}_n)$.

We would not want to read too much into the comparison between the transductive SVM and the NCNM since an exhaustive exploration of the regularisation parameter $C$ was not undertaken. Similar comments also apply to the regularisation of the process variances for the NCNM. However, these preliminary results appear encouraging for the NCNM. Code for recreating all our experiments is available at `http://www.dcs.shef.ac.uk/~neil/ncnm`.

# 6 Discussion

We have presented an approach to learning a classifier in the presence of unlabeled data which incorporates the natural assumption that the data density between classes should be low. Our approach implements this qualitative assumption within a probabilistic framework without explicit, expensive and possibly counterproductive modeling of the class-conditional densities.

Our approach is similar in spirit to the transductive SVM, but with a major difference that in the SVM the process variance is discarded. In the NCNM, the process variance is a key part of data point selection; in particular, Figure 3 illustrated how inclusion of some data points actually increases the posterior process variance. Discarding process variance has advantages and disadvantages—an advantage is that it leads to an optimisation problem that is naturally sparse, while a disadvantage is that it prevents optimisation of kernel parameters via type-II maximum likelihood.

In Section 4.1 we discussed how test data points affect the location of our decision boundary. An important desideratum would be that the location of the decision boundary should converge as the amount of test data goes to infinity. One direction for further research would be to investigate whether or not this is the case.

## Acknowledgments

This work was supported under EPSRC Grant No. GR/R84801/01 and a grant from the National Science Foundation.

## Footnotes

[1]See also [9] who makes use of a similar noise model in a discussion of Bayesian interpretations of the SVM.

[2]Recall, as discussed in Section 1, that we fix the width of the null category to unity: changes in the scale of the process model are equivalent to changing this width.

[3]The informative vector machine is an approximation to a full Gaussian Process which is competitive with the support vector machine in terms of speed and accuracy.

[4]The data set contains 658 examples of 5s and 556 examples of 3s.

[5]Initially we set the value to that learned by the NCNM, but performance was improved by selecting it to be the higher of the two.

# References

[1] A. Agresti. *Categorical Data Analysis*. John Wiley and Sons, 2002.

[2] O. Chapelle, J. Weston, and B. Schölkopf. Cluster kernels for semi-supervised learning. In *Advances in Neural Information Processing Systems*, Cambridge, MA, 2002. MIT Press.

[3] L. Csató. *Gaussian Processes — Iterative Sparse Approximations*. PhD thesis, Aston University, 2002.

[4] T. Joachims. Making large-scale SVM learning practical. In *Advances in Kernel Methods: Support Vector Learning*, Cambridge, MA, 1998. MIT Press.

[5] N. D. Lawrence and B. Schölkopf. Estimating a kernel Fisher discriminant in the presence of label noise. In *Proceedings of the International Conference in Machine Learning*, San Francisco, CA, 2001. Morgan Kaufmann.

[6] N. D. Lawrence, M. Seeger, and R. Herbrich. Fast sparse Gaussian process methods: The informative vector machine. In *Advances in Neural Information Processing Systems*, Cambridge, MA, 2003. MIT Press.

[7] T. P. Minka. *A family of algorithms for approximate Bayesian inference*. PhD thesis, Massachusetts Institute of Technology, 2001.

[8] M. Seeger. Covariance kernels from Bayesian generative models. In *Advances in Neural Information Processing Systems*, Cambridge, MA, 2002. MIT Press.

[9] P. Sollich. Probabilistic interpretation and Bayesian methods for support vector machines. In *Proceedings 1999 International Conference on Artificial Neural Networks, ICANN'99*, pages 91–96, 1999.

[10] V. N. Vapnik. *Statistical Learning Theory*. John Wiley and Sons, New York, 1998.

[11] C. K. I. Williams. Prediction with Gaussian processes: From linear regression to linear prediction and beyond. In *Learning in Graphical Models*, Cambridge, MA, 1999. MIT Press.
